# Bayesian Bias Mitigation for Crowdsourcing

**Fabian L. Wauthier**
University of California, Berkeley
flw@cs.berkeley.edu

**Michael I. Jordan**
University of California, Berkeley
jordan@cs.berkeley.edu

## Abstract

Biased labelers are a systemic problem in crowdsourcing, and a comprehensive toolbox for handling their responses is still being developed. A typical crowdsourcing application can be divided into three steps: data collection, data curation, and learning. At present these steps are often treated separately. We present Bayesian Bias Mitigation for Crowdsourcing (BBMC), a Bayesian model to unify all three. Most data curation methods account for the *effects* of labeler bias by modeling all labels as coming from a single latent truth. Our model captures the *sources* of bias by describing labelers as influenced by shared random effects. This approach can account for more complex bias patterns that arise in ambiguous or hard labeling tasks and allows us to merge data curation and learning into a single computation. Active learning integrates data collection with learning, but is commonly considered infeasible with Gibbs sampling inference. We propose a general approximation strategy for Markov chains to efficiently quantify the effect of a perturbation on the stationary distribution and specialize this approach to active learning. Experiments show BBMC to outperform many common heuristics.

## 1 Introduction

Crowdsourcing is becoming an increasingly important methodology for collecting labeled data, as demonstrated among others by Amazon Mechanical Turk, reCAPTCHA, Netflix, and the ESP game. Motivated by the promise of a wealth of data that was previously impractical to gather, researchers have focused in particular on Amazon Mechanical Turk as a platform for collecting label data [11, 12]. Unfortunately, the data collected from crowdsourcing services is often very dirty: Unhelpful labelers may provide incorrect or biased responses that can have major, uncontrolled effects on learning algorithms. Bias may be caused by personal preference, systematic misunderstanding of the labeling task, lack of interest or varying levels of competence. Further, as soon as malicious labelers try to exploit incentive schemes in the data collection cycle yet more forms of bias enter.

The typical crowdsourcing pipeline can be divided into three main steps: 1) *Data collection*. The researcher farms the labeling tasks to a crowdsourcing service for annotation and possibly adds a small set of gold standard labels. 2) *Data curation*. Since labels from the crowd are contaminated by errors and bias, some filtering is applied to curate the data, possibly using the gold standard provided by the researcher. 3) *Learning*. The final model is learned from the curated data.

At present these steps are often treated as separate. The data collection process is often viewed as a black box which can only be minimally controlled. Although the potential for active learning to make crowdsourcing much more cost effective and goal driven has been appreciated, research on the topic is still in its infancy [4, 9, 17]. Similarly, data curation is in practice often still performed as a preprocessing step, before feeding the data to a learning algorithm [6, 8, 10, 11, 12, 14]. We believe that the lack of systematic solutions to these problems can make crowdsourcing brittle in situations where labelers are arbitrarily biased or even malicious, such as when tasks are particularly ambiguous/hard or when opinions or ratings are solicited.

Our goal in the current paper is to show how crowdsourcing can be leveraged more effectively by treating the overall pipeline within a Bayesian framework. We present Bayesian Bias Mitigation for Crowdsourcing (BBMC) as a way to achieve this. BBMC makes two main contributions.

The first is a flexible latent feature model that describes each labeler's idiosyncrasies through multiple shared factors and allows us to combine data curation and learning (steps 2 and 3 above) into one inferential computation. Most of the literature accounts for the *effects* of labeler bias by assuming a single, true latent labeling from which labelers report noisy observations of some kind [2, 3, 4, 6, 8, 9, 10, 11, 15, 16, 17, 18]. This assumption is inappropriate when labels are solicited on subjective or ambiguous tasks (ratings, opinions, and preferences) or when learning must proceed in the face of arbitrarily biased labelers. We believe that an unavoidable and necessary extension of crowdsourcing allows multiple distinct (yet related) "true" labelings to co-exist, but that at any one time we may be interested in learning about only one of these "truths." Our BBMC framework achieves this by modeling the *sources* of labeler bias through shared random effects.

Next, we want to perform active learning in this model to actively query labelers, thus integrating step 1 with steps 2 and 3. Since our model requires Gibbs sampling for inference, a straightforward application of active learning is infeasible: Each active learning step relies on many inferential computations and would trigger a multitude of subordinate Gibbs samplers to be run within one large Gibbs sampler. Our second contribution is a new methodology for solving this problem. The basic idea is to approximate the stationary distribution of a perturbed Markov chain using that of an unperturbed chain. We specialize this idea to active learning in our model and show that the computations are efficient and that the resulting active learning strategy substantially outperforms other active learning schemes.

The paper is organized as follows: We discuss related work in Section 2. In Section 3 we propose the latent feature model for labelers and in Section 4 we discuss the inference procedure that combines data curation and learning. Then we present a general method to approximate the stationary distribution of perturbed Markov chains and apply it to derive an efficient active learning criterion in Section 5. In Section 6 we present comparative results and we draw conclusions in Section 7.

## 2 Related Work

Relevant work on active learning in multi-teacher settings has been reported in [4, 9, 17]. Sheng et al. [9] use the multiset of current labels with a random forest label model to score which task to next solicit a repeat label for. The quality of the labeler providing the new label does not enter the selection process. In contrast, Donmez et al. [4] actively choose the labeler to query next using a formulation based on interval estimation, utilizing repeated labelings of tasks. The task to label next is chosen separately from the labeler. In contrast, our BBMC framework can perform meaningful inferences even without repeated labelings of tasks and treats the choices of which labeler to query on which task as a joint choice in a Bayesian framework. Yan et al. [17] account for the effects of labeler bias through a coin flip observation model that filters a latent label assignment, which in turn is modeled through a logistic regression. As in [4], the labeler is chosen separately from the task by solving two optimization problems. In other work on data collection strategies, Wais et al. [14] require each labeler to first pass a screening test before they are allowed to label any more data. In a similar manner, reputation systems of various forms are used to weed out historically unreliable labelers before collecting data.

Consensus voting among multiple labels is a commonly used data curation method [12, 14]. It works well when low levels of bias or noise are expected but becomes unreliable when labelers vary greatly in quality [9]. Earlier work on learning from variable-quality teachers was revisited by Smyth et al. [10] who looked at estimating the unknown true label for a task from a set of labelers of varying quality without external gold standard signal. They used an EM strategy to iteratively estimate the true label and the quality of the labelers. The work was extended to a Bayesian formulation by Raykar et al. [8] who assign latent variables to labelers capturing their mislabeling probabilities. Ipeirotis et al. [6] pointed out that a biased labeler who systematically mislabels tasks is still more useful than a labeler who reports labels at random. A method is proposed that separates low quality labelers from high quality, but biased labelers. Dekel and Shamir [3] propose a two-step process. First, they filter labelers by how far they disagree from an estimated true label and then retrain the model on the cleaned data. They give a generalization analysis for anticipated performance. In a

similar vein, Dekel and Shamir [2] show that, under some assumptions, restricting each labeler's influence on a learned model can control the effect of low quality or malicious labelers. Together with [8, 16, 18], [2] and [3] are among the recent lines of research to combine data curation and learning. Work has also focused on using gold standard labels to determine labeler quality. Going beyond simply counting tasks on which labelers disagree with the gold standard, Snow et al. [11] estimate labeler quality in a Bayesian setting by comparing to the gold standard.

Lastly, collaborative filtering has looked extensively at completing sparse matrices of ratings [13]. Given some gold standard labels, collaborative filtering methods could in principle also be used to curate data represented by a sparse label matrix. However, collaborative filtering generally does not combine this inference with the learning of a labeler-specific model for prediction (step 3). Also, with the exception of [19], active learning has not been studied in the collaborative filtering setting.

## 3 Modeling Labeler Bias

In this section we specify a Bayesian latent feature model that accounts for labeler bias and allows us to combine data curation and learning into a single inferential calculation. For ease of exposition we will focus on binary classification, but our method can be generalized. Suppose we solicited labels for $n$ tasks from $m$ labelers. In practical settings it is unlikely that a task is labeled by more than 3–10 labelers [14]. Let task descriptions $x_i \in \mathbb{R}^d$, $i = 1, \ldots, n$, be collected in the matrix $X$. The label responses are recorded in the matrix $Y$ so that $y_{i,l} \in \{-1, 0, +1\}$ denotes the label given to task $i$ by labeler $l$. The special label 0 denotes that a task was not labeled. A researcher is interested in learning a model that can be used to predict labels for new tasks. When consensus is lacking among labelers, our desideratum is to predict the labels that the researcher (or some other expert) would have assigned, as opposed to labels from an arbitrary labeler in the crowd. In this situation it makes sense to stratify the labelers in some way. To facilitate this, the researcher $r$ provides gold standard labels in column $r$ of $Y$ to a small subset of the tasks. Loosely speaking, the gold standard allows our model to curate the data by softly combining labels from those labelers whose responses will useful in predicting $r$'s remaining labels. It is important to note that our model is entirely symmetric in the role of the researcher and labelers. If instead we were interested in predicting labels for labeler $l$, we would treat column $l$ as containing the gold standard labels. The researcher $r$ is just another labeler, the only distinction being that we wish to learn a model that predicts $r$'s labels. To simplify our presentation, we will accordingly refer to labelers in the crowd and the researcher occasionally just as "labelers," indexed by $l$, and only use the distinguishing index $r$ when necessary. We account for each labeler $l$'s idiosyncrasies by assigning a parameter $\beta_l \in \mathbb{R}^d$ to $l$ and modeling labels $y_{i,l}$, $i = 1, \ldots, n$, through a probit model $p(y_{i,l}|x_i, \beta_l) = \Phi(y_{i,l} x_i^\top \beta_l)$, where $\Phi(\cdot)$ is the standard normal CDF. This section describes a joint Bayesian prior on parameters $\beta_l$ that allows for parameter sharing; two labelers that share parameters have similar responses. In the context of this model, the two-step process of data curation and learning a model that predicts $r$'s labels is reduced to posterior inference on $\beta_r$ given $X$ and $Y$. Inference softly integrates labels from relevant labelers, while at the same time allowing us to predict $r$'s remaining labels.

### 3.1 Latent feature model

Labelers are not independent, so it makes sense to impose structure on the set of $\beta_l$'s. Specifically, each vector $\beta_l$ is modeled as the sum of a set of latent factors that are shared across the population. Let $z_l$ be a latent binary vector for labeler $l$ whose component $z_{l,b}$ indicates whether the latent factor $\gamma_b \in \mathbb{R}^d$ contributes to $\beta_l$. In principle, our model allows for an infinite number of distinct factors (i.e., $z_l$ is infinitely long), as long as only a finite number of those factors is active (i.e., $\sum_{b=1}^\infty z_{l,b} < \infty$). Let $\gamma = (\gamma_b)_{b=1}^\infty$ be the concatenation of the factors $\gamma_b$. Given a labeler's vector $z_l$ and factors $\gamma$ we define the parameter $\beta_l = \sum_{b=1}^\infty z_{l,b} \gamma_b$.

For multiple labelers we let the infinitely long matrix $Z = (z_1, \ldots, z_m)^\top$ collect the vectors $z_l$ and define the index set of all observed labels $L = \{(i,l) : y_{i,l} \neq 0\}$, so that the likelihood is

$$p(Y|X, \gamma, Z) = \prod_{(i,l) \in L} p(y_{i,l}|x_i, \gamma, z_l) = \prod_{(i,l) \in L} \Phi(y_{i,l} x_i^\top \beta_l). \tag{1}$$

To complete the model we need to specify priors for $\gamma$ and $Z$. We define the prior distribution of each $\gamma_b$ to be a zero-mean Gaussian $\gamma_b \sim \mathcal{N}(0, \sigma^2 I)$, and let $Z$ be governed by an Indian Buffet

Process (IBP) $Z \sim \text{IBP}(\alpha)$, parameterized by $\alpha$ [5]. The IBP is a stochastic process on infinite binary matrices consisting of vectors $z_l$. A central property of the IBP is that with probability one, a sampled matrix $Z$ contains only a finite number of nonzero entries, thus satisfying our requirement that $\sum_{b=1}^{\infty} z_{l,b} < \infty$. In the context of our model this means that when working with finite data, with probability one only a finite set of features is active across all labelers. To simplify notation in subsequent sections, we use this observation and collapse an infinite matrix $Z$ and vector $\gamma$ to finite dimensional equivalents. From now on, we think of $Z$ as the finite matrix having all zero-columns removed. Similarly, we think of $\gamma$ as having all blocks $\gamma_b$ corresponding to zero-columns in the original matrix $Z$ removed. With probability one, the number of columns $K(Z)$ of $Z$ is finite so we may write $\beta_l = \sum_{b=1}^{K(Z)} z_{l,b} \gamma_b \triangleq Z_l^\top \gamma$, with $Z_l = z_l \otimes I$ the Kronecker product of $z_l$ and $I$.

## 4 Inference: Data Curation and Learning

We noted before that our model combines data curation and learning in a single inferential computation. In this section we lay out the details of a Gibbs sampler for achieving this. Given a task $j$ which was not labeled by $r$ (and possibly no other labeler), we need the predictive probability

$$p(y_{j,r} = +1 | X, Y) = \int p(y_{j,r} = +1 | x_j, \beta_r) p(\beta_r | X, Y) d\beta_r. \tag{2}$$

To approximate this probability we need to gather samples from the posterior $p(\beta_r | Y, X)$. Equivalently, since $\beta_r = Z_r^\top \gamma$, we need samples from the posterior $p(\gamma, z_r | Y, X)$. Because latent factors can be shared across multiple labelers, the posterior will softly absorb label information from labelers whose latent factors tend to be similar to those of the researcher $r$. Thus, Bayesian inference $p(\beta_r | Y, X)$ automatically combines data curation and learning by weighting label information through an inferred sharing structure. Importantly, the posterior is informative even when no labeler in the crowd labeled any of the tasks the researcher labeled.

### 4.1 Gibbs sampling

For Gibbs sampling in the probit model one commonly augments the likelihood in Eq. (1) with intermediate random variables $T = \{t_{i,l} : y_{i,l} \neq 0\}$. The generative model for the label $y_{i,l}$ given $x_i, \gamma$ and $z_l$ first samples $t_{i,l}$ from a Gaussian $\mathcal{N}(\beta_l^\top x_i, 1)$. Conditioned on $t_{i,l}$, the label is then defined as $y_{i,l} = 2\mathbf{1}[t_{i,l} > 0] - 1$. Figure 1(a) summarizes the augmented graphical model by letting $\beta$ denote the collection of $\beta_l$ variables. We are interested in sampling from $p(\gamma, z_r | Y, X)$. The Gibbs sampler for this lives in the joint space of $T, \gamma, Z$ and samples iteratively from the three conditional distributions $p(T | X, \gamma, Z), p(\gamma | X, Z, T)$ and $p(Z | \gamma, X, Y)$. The different steps are:

**Sampling $T$ given $X, \gamma, Z$:** We independently sample elements of $T$ given $X, \gamma, Z$ from a truncated normal as

$$(t_{i,l} | X, \gamma, Z) \sim \mathcal{N}^{y_{i,l}}(t_{i,l} | \gamma^\top Z_l x_i, 1), \tag{3}$$

where we use $\mathcal{N}^{-1}(t | \mu, 1)$ and $\mathcal{N}^{+1}(t | \mu, 1)$ to indicate the density of the negative- and positive-orthant-truncated normal with mean $\mu$ and variance 1, respectively, evaluated at $t$.

**Sampling $\gamma$ given $X, Z, T$:** Straightforward calculations show that conditional sampling of $\gamma$ given $X, Z, T$ follows a multivariate Gaussian

$$(\gamma | X, Z, T) \sim \mathcal{N}(\gamma | \mu, \Sigma), \tag{4}$$

where

$$\Sigma^{-1} = \frac{I}{\sigma^2} + \sum_{(i,l) \in L} Z_l x_i x_i^\top Z_l^\top \qquad \mu = \Sigma \sum_{(i,l) \in L} Z_l x_i t_{i,l}. \tag{5}$$

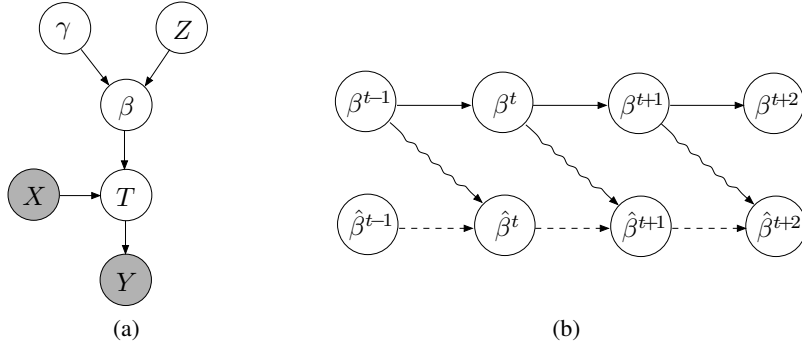

<center>(a)                                                            (b)</center>

Figure 1: (a) A graphical model of the augmented latent feature model. Each node corresponds to a collection of random variables in the model. (b) A schematic of our approximation scheme. The top chain indicates an unperturbed Markov chain, the lower a perturbed Markov chain. Rather than sampling from the lower chain directly (dashed arrows), we transform samples from the top chain to approximate samples from the lower (wavy arrows).

**Sampling $Z$ given $\gamma, X, Y$:** Finally, for inference on $Z$ given $\gamma, X, Y$ we may use techniques outlined in [5]. We are interested in performing active learning in our model, so it is imperative to keep the conditional sampling calculations as compact as possible. One simple way to achieve this is to work with a finite-dimensional approximation to the IBP: We constrain $Z$ to be an $m \times K$ matrix, assigning each labeler at most $K$ active latent features. This is not a substantial limitation; in practice the truncated IBP often performs comparably, and for $K \to \infty$ converges in distribution to the full IBP [5]. Let $m_{-l,b} = \sum_{l' \neq l} z_{l',b}$ be the number of labelers, excluding $l$, with feature $b$ active. Define $\beta_l(z_{l,b}) = z_{l,b}\gamma_b + \sum_{b' \neq b} z_{l,b'}\gamma_{b'}$ as the parameter $\beta_l$ either specifically including or excluding $\gamma_b$. Now if we let $z_{-l,b}$ be the column $b$ of $Z$, excluding element $z_{l,b}$ then updated elements of $Z$ can be sampled one by one as

$$p(z_{l,b} = 1|z_{-l,b}) = \frac{m_{-l,b} + \frac{\alpha}{K}}{n + \frac{\alpha}{K}} \tag{6}$$

$$p(z_{l,b}|z_{-l,b}, \gamma, X, Y) \propto p(z_{l,b}|z_{-l,b}) \prod_{i:y_{i,l} \neq 0} \Phi(y_{i,l}x_i^\top \beta_l(z_{l,b})). \tag{7}$$

After reaching approximate stationarity, we collect samples $(\gamma^s, Z^s)$, $s = 1, \ldots, S$, from the Gibbs sampler as they are generated. We then compute samples from $p(\beta_r|Y, X)$ by writing $\beta_r^s = Z_r^{s\top}\gamma^s$.

## 5   Active Learning

The previous section outlined how, given a small set of gold standard labels from $r$, the remaining labels can be predicted via posterior inference $p(\beta_r|Y, X)$. In this section we take an active learning approach [1, 7] to incrementally add labels to $Y$ so as to quickly learn about $\beta_r$ while reducing data acquisition costs. Active learning allows us to guide the data collection process through model inferences, thus integrating the *data collection*, *data curation* and *learning* steps of the crowdsourcing pipeline. We envision a unified system that automatically asks for more labels from those labelers on those tasks that are most useful in inferring $\beta_r$. This is in contrast to [9], where labelers cannot be targeted with tasks. It is also unlike [4] since we can let labelers be arbitrarily unhelpful, and differs from [17] which assumes a single latent truth.

A well-known active learning criterion popularized by Lindley [7] is to label that task next which maximizes the prior-posterior reduction in entropy of an inferential quantity of interest. The original formulation has been generalized beyond entropy to arbitrary utility functionals $U(\cdot)$ of the updated posterior probability [1]. The functional $U(\cdot)$ is a model parameter that can depend on the type of inferences we are interested in. In our particular setup, we wish to infer the parameter $\beta_r$ to predict labels for the researcher $r$. Suppose we chose to solicit a label for task $i'$ from labeler $l'$, which produced label $y_{i',l'}$. The utility of this observation is $U(p(\beta_r|y_{i',l'}))$. The average utility of receiving a label on task $i'$ from labeler $l'$ is $\mathcal{I}((i', l'), p(\beta_r)) = E(U(p(\beta_r|y_{i',l'})))$, where the expectation is taken with respect to the predictive label probabilities $p(y_{i',l'}|x_{i'}) = \int p(y_{i',l'}|x_{i'}, \beta_{l'})p(\beta_{l'})d\beta_{l'}$. Active learning chooses that pair $(i', l')$ which maximizes $\mathcal{I}((i', l'), p(\beta_r))$. If we want to choose the next task for the researcher to label, we constrain $l' = r$. To query the crowd we let $l' \neq r$. Similarly, we can constrain $i'$ to any particular value or subset of interest. For the following discussion we let $U(p(\beta_r|y_{i',l'})) = \|E_{p(\beta_r)}(\beta_r) - E_{p(\beta_r|y_{i',l'})}(\beta_r)\|_2$ be the $\ell_2$ norm of the difference in means of $\beta_r$. Picking the task that shifts the posterior mean the most is similar in spirit to the common criterion of maximizing the Kullback-Leibler divergence between the prior and posterior.

## 5.1 Active learning for MCMC inference

A straightforward application of active learning is impractical using Gibbs sampling, because to score a single task-labeler pair $(i', l')$ we would have to run two Gibbs samplers (one for each of the two possible labels) in order to approximate the updated posterior distributions. Suppose we started with $k$ task-labeler pairs that active learning could choose from. Depending on the number of selections we wish to perform, we would have to run $k \lesssim g \lesssim k^2$ Gibbs samplers *within* the topmost Gibbs sampler of Section 4. Clearly, such a scoring approach is not practical. To solve this problem, we propose a general purpose strategy to approximate the stationary distribution of a perturbed Markov chain using that of an unperturbed Markov chain. The approximation allows efficient active learning in our model that outperforms naïve scoring both in speed and quality.

The main idea can be summarized as follows. Suppose we have two Markov chains, $p(\beta_r^t|\beta_r^{t-1})$ and $\hat{p}(\hat{\beta}_r^t|\hat{\beta}_r^{t-1})$, the latter of which is a slight perturbation of the former. Denote the stationary distributions by $p_\infty(\beta_r)$ and $\hat{p}_\infty(\hat{\beta}_r)$, respectively. If we are given the stationary distribution $p_\infty(\beta_r)$ of the unperturbed chain, then we propose to approximate the perturbed stationary distribution by

$$\hat{p}_\infty(\hat{\beta}_r) \approx \int \hat{p}(\hat{\beta}_r|\beta_r)p_\infty(\beta_r)d\beta_r. \tag{8}$$

If $\hat{p}(\hat{\beta}^t|\hat{\beta}^{t-1}) = p(\hat{\beta}^t|\hat{\beta}^{t-1})$ the approximation is exact. Our hope is that if the perturbation is small enough the above approximation is good. To use this practically with MCMC, we first run the unperturbed MCMC chain to approximate stationarity, and then use samples of $p_\infty(\beta_r)$ to compute approximate samples from $\hat{p}_\infty(\hat{\beta}_r)$. Figure 1(b) shows this scheme visually.

To map this idea to our active learning setup we conceptually let the unperturbed chain $p(\beta_r^t|\beta_r^{t-1})$ be the chain on $\beta_r$ induced by the Gibbs sampler in Section 4. The perturbed chain $\hat{p}(\hat{\beta}_r^t|\hat{\beta}_r^{t-1})$ represents the chain where we have added a new observation $y_{i',l'}$ to the measured data. If we have $S$ samples $\beta_r^s$ from $p_\infty(\beta_r)$, then we approximate the perturbed distribution as

$$\hat{p}_\infty(\hat{\beta}_r) \approx \frac{1}{S}\sum_{s=1}^{S} \hat{p}(\hat{\beta}_r|\beta_r^s), \tag{9}$$

and the active learning score as $U(p(\beta_r|y_{i',l'})) \approx U\left(\hat{p}_\infty(\hat{\beta}_r)\right)$. To further specialize this strategy to our model we first rewrite the Gibbs sampler outlined in Section 4. We suppress mentions of $X$ and $Y$ in the subsequent presentation. Instead of first sampling $(T|\gamma^{t-1}, Z)$ from Eq. (3), and then sampling $(\gamma^t|T, Z)$ from Eq. (4), we combine them into one larger sampling step $(\gamma^t|\gamma^{t-1}, Z)$. Starting from a fixed $\gamma^{t-1}$ and $Z$ we sample from $\gamma^t$ as

$$\left(\gamma^t|\gamma^{t-1}, Z\right) \overset{d}{=} \eta_\Sigma + \mu = \Sigma \left[\eta_{\sigma^{-2}I} + \sum_{(i,l)\in L} Z_l x_i \left[\eta_1 + \left(t_{i,l}|\gamma^{t-1}, Z\right)\right]\right], \tag{10}$$

where $\eta_\Sigma$ is a zero-mean Gaussian with covariance $\Sigma$, and $\eta_1$ a standard normal random variable. If it were feasible, we could also absorb the intermediate sampling of $Z$ into the notation and write down a single induced Markov chain $(\beta_r^t|\beta_r^{t-1})$, as referred to in Eqs. (8) and (9). As this is not possible, we will account for $Z$ separately. We see that the effect of adding a new observation $y_{i',l'}$ is to perturb the Markov chain in Eq. (10) by adding an element to $L$. Supposing we added this new observation at time $t-1$, let $\Sigma_{(i',l')}$ be defined as $\Sigma$ but with $(i', l')$ added to $L$. Straightforward calculations using the Sherman-Morrison-Woodbury identity on $\Sigma_{(i',l')}$ give that, conditioned on $\gamma^{t-1}, Z$, we can

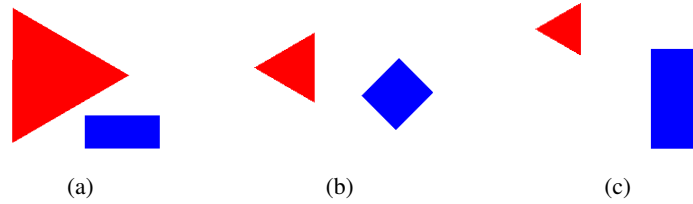

(a)  (b)  (c)

Figure 2: Examples of easy and ambiguous labeling tasks. We asked labelers to determine if the triangle is to the left or above the square.

write the first step of the perturbed Gibbs sampler as a function of the unperturbed Gibbs sampler. If we let $A_{i',l'} = \Sigma Z_{l'} x_{i'} x_{i'}^\top Z_{l'}^\top / (1 + x_{i'}^\top Z_{l'}^\top \Sigma Z_{l'} x_{i'})$ for compactness, then we yield

$$\left(\gamma_{(i',l')}^t | \gamma^{t-1}, Z\right) \overset{d}{=} (I - A_{i',l'}) \left(\gamma^t | \gamma^{t-1}, Z\right) + \Sigma_{(i',l')} Z_{l'} x_{i'} \left[\eta_1 + \left(t_{i',l'} | \gamma^{t-1}, Z\right)\right]. \quad (11)$$

To approximate the utility $U(\cdot)$ we now appeal to Eq. (9) and estimate the difference in means using recent samples $\gamma^s, Z^s, s = 1, \ldots, S$ from the unperturbed sampler. In terms of Eqs. (10) and (11),

$$U(p(\beta_r | y_{i',l'})) = \left\| E_{p(\beta_r)}(\beta_r) - E_{p(\beta_r | y_{i',l'})}(\beta_r) \right\|_2 \quad (12)$$

$$\approx \left\| E \left( \frac{1}{S-1} \sum_{s=2}^S Z_r^{s-1^\top} \left[ \left(\gamma | \gamma^{s-1}, Z^{s-1}\right) - \left(\gamma_{(i',l')} | \gamma^{s-1}, Z^{s-1}\right) \right] \right) \right\|_2. \quad (13)$$

By simple cancellations and expectations of truncated normal variables we can reduce the above expression to a sample average of elementary calculations. Note that the sample $\gamma^s$ is a realization of $\left(\gamma | \gamma^{s-1}, Z^{s-1}\right)$. We have used this to approximate $E\left(\left(\gamma | \gamma^{s-1}, Z^{s-1}\right)\right) \approx \gamma^s$. Thus, the sum only runs over $S-1$ terms. In principle the exact expectation could also be computed. The final utility calculation is straightforward but too long to expand. Finally, we use samples from the Gibbs sampler to approximate $p(y_{i',l'} | x_{i'})$ and estimate $\mathcal{I}((i', l'), p(\beta_r))$ for querying labeler $l'$ on task $i'$.

## 6 Experimental Results

We evaluated our active learning method on an ambiguous localization task which asked labelers on Amazon Mechanical Turk to determine if a triangle was to the left or above a rectangle. Examples are shown in Figure 6. Tasks such as these are important for learning computer vision models of perception. Rotation, translation and scale, as well as aspect ratios, were pseudo-randomly sampled in a way that produced ambiguous tasks. We expected labelers to use centroids, extreme points and object sizes in different ways to solve the tasks, thus leading to structurally biased responses. Additionally, our model will also have to deal with other forms of noise and bias. The gold standard was to compare only the centroids of the two objects. For training we generated 1000 labeling tasks and solicited 3 labels for each task. Tasks were solved by 75 labelers with moderate disagreement. To emphasize our results, we retained only the subset of 523 tasks with disagreement. We provided about 60 gold standard labels to BBMC and then performed inference and active learning on $\beta_r$ so as to learn a predictive model emulating gold standard labels. We evaluated methods based on the log likelihood and error rate on a held-out test set of 1101 datapoints.[1] All results shown in Table 1 were averaged across 10 random restarts. We considered two scenarios. The first compares our model to other methods when no active learning is performed. This will demonstrate the advantages of the latent feature model presented in Sections 3 and 4. The second scenario compares performance of our active learning scheme to various other methods. This will highlight the viability of our overall scheme presented in Section 5 that ties data collection together with data curation and learning.

First we show performance without active learning. Here only about 60 gold standard labels and all the labeler data is available for training. The results are shown in the top three rows of Table 1. Our method, "BBMC," outperforms the other two methods by a large margin. The BBMC scores were computed by running the Gibbs sampler of Section 4 with 2000 iterations burnin and then computing

|          | Final Loglik       | Final Error           |
|----------|--------------------|-----------------------|
| GOLD     | $-3716 \pm 1695$   | $0.0547 \pm 0.0102$   |
| CONS     | $-421.1 \pm 2.6$   | $0.0935 \pm 0.0031$   |
| BBMC     | $\mathbf{-219.1 \pm 3.1}$ | $\mathbf{0.0309 \pm 0.0033}$ |
| GOLD-ACT | $-1957 \pm 696$    | $0.0290 \pm 0.0037$   |
| CONS-ACT | $-396.1 \pm 3.6$   | $0.0906 \pm 0.0024$   |
| RAND-ACT | $-186.0 \pm 2.2$   | $0.0292 \pm 0.0029$   |
| DIS-ACT  | $-198.3 \pm 5.8$   | $0.0392 \pm 0.0052$   |
| MCMC-ACT | $-196.1 \pm 6.7$   | $0.0492 \pm 0.0050$   |
| BBMC-ACT | $\mathbf{-160.8 \pm 3.9}$ | $\mathbf{0.0188 \pm 0.0018}$ |

Table 1: The top three rows give results without and the bottom six rows results with active learning.

a predictive model by averaging over the next 20000 iterations. The alternatives include "GOLD," which is a logistic regression trained only on gold standard labels, and "CONS," which evaluates logistic regression trained on the overall majority consensus. Training on the gold standard only often overfits, and training on the consensus systematically misleads.

Next, we evaluate our active learning method. As before, we seed the model with about 60 gold standard labels. We repeatedly select a new task for which to receive a gold standard label from the researcher. That is, for this experiment we constrained active learning to use $l' = r$. Of course, in our framework we could have just as easily queried labelers in the crowd. Following 2000 steps burnin we performed active learning every 200 iterations for a total of 100 selections. The reported scores were computed by estimating a predictive model from the last 200 iterations. The results are shown in the lower six rows of Table 1. Our model with active learning, "BBMC-ACT," outperforms all alternatives. The first alternative we compared against, "MCMC-ACT," does active learning with the MCMC-based scoring method outlined in Section 5. In line with our utility $U(\cdot)$ this method scores a task by running two Gibbs samplers within the overall Gibbs sampler and then approximates the expected mean difference of $\beta_r$. Due to time constraints, we could only afford to run each subordinate chain for 10 steps. Even then, this method requires on the order of $10 \times 83500$ Gibbs sampling iterations for 100 active learning steps. It takes about 11 hours to run the entire chain, while BBMC only requires 2.5 hours. The MCMC method performs very poorly. This demonstrates our point: Since the MCMC method computes a similar quantity as our approximation, it should perform similarly given enough iterations in each subchain. However, 10 iterations is not nearly enough time for the scoring chains to mix and also quite a small number to compute empirical averages, leading to decreased performance. A more realistic alternative to our model is "DIS-ACT," which picks one of the tasks with most labeler disagreement to label next. Lastly, the baseline alternatives include "GOLD-ACT" and "CONS-ACT" which pick a random task to label and then learn logistic regressions on the gold standard or consensus labels respectively. Those results can be directly compared against "RAND-ACT," which uses our model and inference procedure but similarly selects tasks at random. In line with our earlier evaluation, we still outperform these two methods when effectively no active learning is done.

## 7 Conclusions

We have presented Bayesian Bias Mitigation for Crowdsourcing (BBMC) as a framework to unify the three main steps in the crowdsourcing pipeline: data collection, data curation and learning. Our model captures labeler bias through a flexible latent feature model and conceives of the entire pipeline in terms of probabilistic inference. An important contribution is a general purpose approximation strategy for Markov chains that allows us to efficiently perform active learning, despite relying on Gibbs sampling for inference. Our experiments show that BBMC is fast and greatly outperforms a number of commonly used alternatives.

### Acknowledgements

We would like to thank Purnamrita Sarkar for helpful discussions and Dave Golland for assistance in developing the Amazon Mechanical Turk HITs.

## Footnotes

[1]The test set was similarly constructed by selecting from 2000 tasks those on which three labelers disagreed.

# References

[1] K. Chaloner and I. Verdinelli. Bayesian Experimental Design: A Review. *Statistical Science*, 10(3):273–304, 1995.

[2] O. Dekel and O. Shamir. Good Learners for Evil Teachers. In L. Bottou and M. Littman, editors, *Proceedings of the 26th International Conference on Machine Learning (ICML)*. Omnipress, 2009.

[3] O. Dekel and O. Shamir. Vox Populi: Collecting High-Quality Labels from a Crowd. In *Proceedings of the 22nd Annual Conference on Learning Theory (COLT)*, Montreal, Quebec, Canada, 2009.

[4] P. Donmez, J. G. Carbonell, and J. Schneider. Efficiently Learning the Accuracy of Labeling Sources for Selective Sampling. In *Proceedings of the 15th ACM SIGKDD*, KDD, Paris, France, 2009.

[5] T. L. Griffiths and Z. Ghahramani. Infinite Latent Feature Models and the Indian Buffet Process. Technical report, Gatsby Computational Neuroscience Unit, 2005.

[6] P. G. Ipeirotis, F. Provost, and J. Wang. Quality Management on Amazon Mechanical Turk. In *Proceedings of the ACM SIGKDD Workshop on Human Computation*, HCOMP, pages 64–67, Washington DC, 2010.

[7] D. V. Lindley. On a Measure of the Information Provided by an Experiment. *The Annals of Mathematical Statistics*, 27(4):986–1005, 1956.

[8] V. C. Raykar, S. Yu, L. H. Zhao, G. H. Valadez, C. Florin, L. Bogoni, and L. Moy. Learning from Crowds. *Journal of Machine Learning Research*, 11:1297–1322, April 2010.

[9] V. S. Sheng, F. Provost, and P. G. Ipeirotis. Get Another Label? Improving Data Quality and Data Mining using Multiple, Noisy Labelers. In *Proceeding of the 14th ACM SIGKDD*, KDD, Las Vegas, Nevada, 2008.

[10] P. Smyth, U. M. Fayyad, M. C. Burl, P. Perona, and P. Baldi. Inferring Ground Truth from Subjective Labelling of Venus Images. In G. Tesauro, D. S. Touretzky, and T. K. Leen, editors, *Advances in Neural Information Processing Systems 7 (NIPS)*. MIT Press, 1994.

[11] R. Snow, B. O'Connor, D. Jurafsky, and A. Y. Ng. Cheap and Fast—But is it Good? Evaluating Non-Expert Annotations for Natural Language Tasks. In *Proceedings of EMNLP*. Association for Computational Linguistics, 2008.

[12] A. Sorokin and D. Forsyth. Utility Data Annotation with Amazon Mechanical Turk. In *CVPR Workshop on Internet Vision*, Anchorage, Alaska, 2008.

[13] X. Su and T. M. Khoshgoftaar. A Survey of Collaborative Filtering Techniques. *Advances in Artificial Intelligence*, 2009:4:2–4:2, January 2009.

[14] P. Wais, S. Lingamnei, D. Cook, J. Fennell, B. Goldenberg, D. Lubarov, D. Marin, and H. Simons. Towards Building a High-Quality Workforce with Mechanical Turk. In *NIPS Workshop on Computational Social Science and the Wisdom of Crowds*, Whistler, BC, Canada, 2010.

[15] P. Welinder, S. Branson, S. Belongie, and P. Perona. The Multidimensional Wisdom of Crowds. In J. Lafferty, C. K. I. Williams, R. Zemel, J. Shawe-Taylor, and A. Culotta, editors, *Advances in Neural Information Processing Systems 23 (NIPS)*. MIT Press, 2010.

[16] J. Whitehill, P. Ruvolo, T. Wu, J. Bergsma, and J. Movellan. Whose Vote Should Count More: Optimal Integration of Labels from Labelers of Unknown Expertise. In Y. Bengio, D. Schuurmans, J. Lafferty, C. K. I. Williams, and A. Culotta, editors, *Advances in Neural Information Processing Systems 22 (NIPS)*. MIT Press, 2009.

[17] Y. Yan, R. Rosales, G. Fung, and J. G. Dy. Active Learning from Crowds. In L. Getoor and T. Scheffer, editors, *Proceedings of the 28th International Conference on Machine Learning (ICML)*, Bellevue, Washington, 2011.

[18] Y. Yan, R. Rosales, G. Fung, M. Schmidt, G. Hermosillo, L. Bogoni, L. Moy, and J. G. Dy. Modeling Annotator Expertise: Learning When Everybody Knows a Bit of Something. In *Proceedings of AISTATS*, volume 9, Chia Laguna, Sardinia, Italy, 2010.

[19] K. Yu, A. Schwaighofer, V. Tresp, X. Xu, and H. Kriegel. Probabilistic Memory-based Collaborative Filtering. *IEEE Transactions On Knowledge and Data Engineering*, 16(1):56–69, January 2004.

